# A joint maximum-entropy model for binary neural population patterns and continuous signals

**Sebastian Gerwinn**       **Philipp Berens**       **Matthias Bethge**
MPI for Biological Cybernetics
and University of Tübingen
Computational Vision and Neuroscience
Spemannstrasse 41, 72076 Tübingen, Germany
`{firstname.surname}@tuebingen.mpg.de`

## Abstract

Second-order maximum-entropy models have recently gained much interest for describing the statistics of binary spike trains. Here, we extend this approach to take continuous stimuli into account as well. By constraining the joint second-order statistics, we obtain a joint Gaussian-Boltzmann distribution of continuous stimuli and binary neural firing patterns, for which we also compute marginal and conditional distributions. This model has the same computational complexity as pure binary models and fitting it to data is a convex problem. We show that the model can be seen as an extension to the classical spike-triggered average/covariance analysis and can be used as a non-linear method for extracting features which a neural population is sensitive to. Further, by calculating the posterior distribution of stimuli given an observed neural response, the model can be used to decode stimuli and yields a natural spike-train metric. Therefore, extending the framework of maximum-entropy models to continuous variables allows us to gain novel insights into the relationship between the firing patterns of neural ensembles and the stimuli they are processing.

## 1  Introduction

Recent technical advances in systems neuroscience allow us to monitor the activity of increasingly large neural ensembles simultaneously (e.g. [5, 21]). To understand how such ensembles process sensory information and perform the complex computations underlying successful behavior requires not only collecting massive amounts of data, but also the use of suitable statistical models for data analysis. What degree of precision should be incorporated into such a model involves a trade-off between the question of interest and mathematical tractability: Complex multi-compartmental models [8] allow inference concerning the underlying biophysical processes, but their applicability to neural populations is limited. The generalized linear model [15] on the other hand is tractable even for large ensembles and provides a phenomenological description of the data.

Recently, several groups have used binary maximum entropy models incorporating pairwise correlations to model neural activity in large populations of neurons on short time scales [19, 22, 7, 25]. These models have two important features: (1) Since they only require measuring the mean activity of individual neurons and correlations in pairs of neurons, they can be estimated from moderate amounts of data. (2) They seem to capture the essential structure of neural population activity at these timescales even in networks of up to a hundred neurons [21]. Although the generality of these findings have been subject to debate [3, 18], pairwise maximum-entropy and related models [12] are an important tool for the description of neural population activity [23, 17].

To find features to which a neuron is sensitive spike-triggered average and spike-triggered covariance are commonly used techniques [20, 16]. They correspond to fitting a Gaussian distribution to the spike-triggered ensemble. If one has access to multi-neuron recordings, a straightforward extension of this approach is to fit a different Gaussian distribution to each binary population pattern. In statistics, the corresponding model is known as the location model [14, 10, 9]. To estimate this model, one has to observe sufficient amounts of data for each population pattern. As the number of possible binary patterns grows exponentially with the number of neurons, it is desirable to include regularization constraints in order to make parameter estimation tractable.

Here, we extend the framework of pairwise maximum entropy modeling to a joint model for binary and continuous variables. This allows us to analyze the functional connection structure in a neural population at the same time as its relationship with further continuous signals of interest. In particular, this approach makes it possible to include a stimulus as a continuous variable into the framework of maximum-entropy modeling. In this way, we can study the stimulus dependence of binary neural population activity in a regularized framework in a rigorous way. In particular, we can use it to extract non-linear features in the stimulus that a population of neurons is sensitive to, while taking the binary nature of spike trains into account. We discuss the relationship of the obtained features with classical approaches such as spike-triggered average (STA) and spike-triggered covariance (STC). In addition, we show how the model can be used to perform spike-by-spike decoding and yields a natural spike-train metric [24, 2]. We start with a derivation of the model and a discussion of its features.

## 2 Model

In this section we derive the maximum-entropy model for joint continuous and binary data with second-order constraints and describe its basic properties. We write continuous variables $\mathbf{x}$ and binary variables $\mathbf{b}$. Having observed the joint mean $\boldsymbol{\mu}$ and joint covariance $\mathbf{C}$, we want to find a distribution $p_{\mathrm{ME}}$ which achieves the maximal entropy under all distributions with these observed moments. Since we model continuous and binary variables jointly, we define entropy to be a mixed discrete entropy and differential entropy:

$$H[p] = -\sum_{\mathbf{b}} \int p(\mathbf{x}, \mathbf{b}) \log p(\mathbf{x}, \mathbf{b}) \mathrm{d}\mathbf{x}$$

Formally, we require $p_{\mathrm{ME}}$ to satisfy the following constraints:

$$
\begin{aligned}
\mathbb{E}[\mathbf{x}] &= \boldsymbol{\mu}_x & \mathbb{E}[\mathbf{b}] &= \boldsymbol{\mu}_b \\
\mathbb{E}[\mathbf{x}\mathbf{x}^\top] &= \mathbf{C}_{xx} + \boldsymbol{\mu}_x\boldsymbol{\mu}_x^\top & \mathbb{E}[\mathbf{b}\mathbf{b}^\top] &= \mathbf{C}_{bb} + \boldsymbol{\mu}_b\boldsymbol{\mu}_b^\top \\
\mathbb{E}[\mathbf{x}\mathbf{b}^\top] &= \mathbf{C}_{xb} + \boldsymbol{\mu}_x\boldsymbol{\mu}_b^\top & \mathbb{E}[\mathbf{b}\mathbf{x}^\top] &= \mathbf{C}_{bx} + \boldsymbol{\mu}_b\boldsymbol{\mu}_x^\top = \mathbf{C}_{xb} + \boldsymbol{\mu}_x\boldsymbol{\mu}_b^\top
\end{aligned}
\tag{1}
$$

where the expectations are taken over $p_{\mathrm{ME}}$. $\mathbf{C}_{xx}, \mathbf{C}_{xb}$ and $\mathbf{C}_{bb}$ are blocks in the observed covariance matrix corresponding to the respective subsets of variables. This problem can be solved analytically using the Lagrange formalism, which leads to a maximum entropy distribution of Boltzmann type:

$$p_{\mathrm{ME}}(\mathbf{x}, \mathbf{b} | \boldsymbol{\Lambda}, \boldsymbol{\lambda}) = \frac{1}{Z(\boldsymbol{\Lambda}, \boldsymbol{\lambda})} \exp\left(Q\left(\mathbf{x}, \mathbf{b} | \boldsymbol{\Lambda}, \boldsymbol{\lambda}\right)\right)$$

$$Q(\mathbf{x}, \mathbf{b} | \boldsymbol{\Lambda}, \boldsymbol{\lambda}) = \frac{1}{2} \begin{pmatrix} \mathbf{x} \\ \mathbf{b} \end{pmatrix}^\top \boldsymbol{\Lambda} \begin{pmatrix} \mathbf{x} \\ \mathbf{b} \end{pmatrix} + \boldsymbol{\lambda}^\top \begin{pmatrix} \mathbf{x} \\ \mathbf{b} \end{pmatrix} \tag{2}$$

$$Z(\boldsymbol{\Lambda}, \boldsymbol{\lambda}) = \sum_{\mathbf{b}} \int \exp\left(Q\left(\mathbf{x}, \mathbf{b} | \boldsymbol{\Lambda}, \boldsymbol{\lambda}\right)\right) \mathrm{d}\mathbf{x},$$

where $\boldsymbol{\Lambda}$ and $\boldsymbol{\lambda}$ are chosen such that the resulting distribution fulfills the constraints in equation (1), as we discuss below. Before we compute marginal and conditional distributions in this model, we explore its basic properties. First, we note that the joint distribution can be factorized in the following way:

$$p_{\mathrm{ME}}(\mathbf{x}, \mathbf{b} | \boldsymbol{\Lambda}, \boldsymbol{\lambda}) = p_{\mathrm{ME}}(\mathbf{x} | \mathbf{b}, \boldsymbol{\Lambda}, \boldsymbol{\lambda}) p_{\mathrm{ME}}(\mathbf{b} | \boldsymbol{\Lambda}, \boldsymbol{\lambda}) \tag{3}$$

The conditional density $p_{\text{ME}}(\mathbf{x}|\mathbf{b}, \mathbf{\Lambda}, \boldsymbol{\lambda})$ is a Normal distribution, given by:

$$p_{\text{ME}}(\mathbf{x}|\mathbf{b}, \mathbf{\Lambda}, \boldsymbol{\lambda}) \propto \exp\left(\frac{1}{2}\mathbf{x}^\top \mathbf{\Lambda}_{xx}\mathbf{x} + \mathbf{x}^\top\left(\boldsymbol{\lambda}_x + \mathbf{\Lambda}_{xb}\mathbf{b}\right)\right) \tag{4}$$

$$\propto \mathcal{N}\left(\mathbf{x}|\boldsymbol{\mu}_{x|b}, \mathbf{\Sigma}\right) \quad \text{,with} \quad \boldsymbol{\mu}_{x|b} = \mathbf{\Sigma}\left(\boldsymbol{\lambda}_x + \mathbf{\Lambda}_{xb}\mathbf{b}\right), \qquad \mathbf{\Sigma} = \left(-\mathbf{\Lambda}_{xx}\right)^{-1}$$

Here, $\mathbf{\Lambda}_{xx}, \mathbf{\Lambda}_{xb}, \mathbf{\Lambda}_{bx}, \boldsymbol{\lambda}_x$ are the blocks in $\mathbf{\Lambda}$ which correspond to $\mathbf{x}$ and $\mathbf{b}$, respectively. While the mean of this Normal distribution dependent on $\mathbf{b}$, the covariance matrix is independent of the specific binary state. The marginal probability $p_{\text{ME}}(\mathbf{b}|\mathbf{\Lambda}, \boldsymbol{\lambda})$ is given by:

$$Z(\mathbf{\Lambda}, \boldsymbol{\lambda})p_{\text{ME}}(\mathbf{b}|\Lambda, \lambda) = \exp\left(\frac{1}{2}\mathbf{b}^\top\mathbf{\Lambda}_{bb}\mathbf{b} + \mathbf{b}^\top\boldsymbol{\lambda}_b\right)\int \exp\left(\frac{1}{2}\mathbf{x}^\top\mathbf{\Lambda}_{xx}\mathbf{x} + \mathbf{x}^\top\left(\boldsymbol{\lambda}_x + \mathbf{\Lambda}_{xb}\mathbf{b}\right)\right)\,\mathrm{d}\mathbf{x}$$

$$= (2\pi)^{\frac{n}{2}}\left|-\mathbf{\Lambda}_{xx}\right|^{-\frac{1}{2}}\exp\left(\frac{1}{2}\mathbf{b}^\top\left(\mathbf{\Lambda}_{bb} + \mathbf{\Lambda}_{xb}^\top\left(-\mathbf{\Lambda}_{xx}\right)^{-1}\mathbf{\Lambda}_{xb}\right)\mathbf{b}\right. \tag{5}$$

$$\left.+ \mathbf{b}^\top\left(\boldsymbol{\lambda}_b + \mathbf{\Lambda}_{xb}^\top\left(-\mathbf{\Lambda}_{xx}\right)^{-1}\boldsymbol{\lambda}_x\right) + \frac{1}{2}\boldsymbol{\lambda}_x^\top\left(-\mathbf{\Lambda}_{xx}\right)^{-1}\boldsymbol{\lambda}_x\right)$$

To evaluate the maximum entropy distribution, we need to compute the partition function, which follows from the previous equation by summing over $\mathbf{b}$:

$$Z(\mathbf{\Lambda}, \boldsymbol{\lambda}) = (2\pi)^{\frac{n}{2}}\left|-\mathbf{\Lambda}_{xx}\right|^{-\frac{1}{2}}\sum_b \exp\left(\frac{1}{2}\mathbf{b}^\top\left(\mathbf{\Lambda}_{bb} + \mathbf{\Lambda}_{xb}^\top\left(-\mathbf{\Lambda}_{xx}\right)^{-1}\mathbf{\Lambda}_{xb}\right)\mathbf{b}\right.$$

$$\left.+ \mathbf{b}^\top\left(\boldsymbol{\lambda}_b + \mathbf{\Lambda}_{xb}^\top\left(-\mathbf{\Lambda}_{xx}\right)^{-1}\boldsymbol{\lambda}_x\right) + \frac{1}{2}\boldsymbol{\lambda}_x^\top\left(-\mathbf{\Lambda}_{xx}\right)^{-1}\boldsymbol{\lambda}_x\right) \tag{6}$$

Next, we compute the marginal distribution with respect to $\mathbf{x}$. From equation (5) and (4), we find that $p_{\text{ME}}(\mathbf{x}|\mathbf{\Lambda}, \boldsymbol{\lambda})$ is a mixture of Gaussians, where each Gaussian of equation (4) is weighted by the corresponding $p_{\text{ME}}(\mathbf{b}|\mathbf{\Lambda}, \boldsymbol{\lambda})$. While all mixture components have the same covariance, the different weighting terms affect each component's influence on the marginal covariance of $\mathbf{x}$. Finally, we also compute the conditional density $p_{\text{ME}}(\mathbf{b}|\mathbf{x}, \Lambda, \lambda)$, which is given by:

$$p_{\text{ME}}(\mathbf{b}|\mathbf{x}, \mathbf{\Lambda}, \boldsymbol{\lambda}) = \frac{1}{Z'}\exp\left(\frac{1}{2}\mathbf{b}^\top\mathbf{\Lambda}_{bb}\mathbf{b} + \mathbf{b}^\top\left(\boldsymbol{\lambda}_b + \mathbf{\Lambda}_{bx}\mathbf{x}\right)\right)$$

$$Z' = \sum_\mathbf{b}\exp\left(\frac{1}{2}\mathbf{b}^\top\mathbf{\Lambda}_{bb}\mathbf{b} + \mathbf{b}^\top\left(\boldsymbol{\lambda}_b + \mathbf{\Lambda}_{bx}\mathbf{x}\right)\right) \tag{7}$$

Note, that the distribution of the binary variables given the continuous variables is again of Boltzmann type.

**Parameter fitting** To find suitable parameters for given data, we employ a maximum likelihood approach [1, 11], where we find the optimal parameters via gradient descent:

$$l(\mathbf{\Lambda}, \boldsymbol{\lambda}) = \log p(\{\mathbf{x}^{(n)}, \mathbf{b}^{(n)}\}_{n=1}^N|\mathbf{\Lambda}, \boldsymbol{\lambda}) = \sum_n Q(\mathbf{x}^{(n)}, \mathbf{b}^{(n)}|\Lambda, \lambda) - N\log Z\left(\mathbf{\Lambda}, \boldsymbol{\lambda}\right) \tag{8}$$

$$\Rightarrow \nabla_{\mathbf{\Lambda}} l = N\left[\left\langle\begin{pmatrix}\mathbf{x}\\\mathbf{b}\end{pmatrix}\begin{pmatrix}\mathbf{x}\\\mathbf{b}\end{pmatrix}^\top\right\rangle_{\text{data}} - \left\langle\begin{pmatrix}\mathbf{x}\\\mathbf{b}\end{pmatrix}\begin{pmatrix}\mathbf{x}\\\mathbf{b}\end{pmatrix}^\top\right\rangle_{p_{\text{ME}}}\right]$$

$$\nabla_{\boldsymbol{\lambda}} l = N\left[\left\langle\begin{pmatrix}\mathbf{x}\\\mathbf{b}\end{pmatrix}\right\rangle_{\text{data}} - \left\langle\begin{pmatrix}\mathbf{x}\\\mathbf{b}\end{pmatrix}\right\rangle_{p_{\text{ME}}}\right]$$

To calculate the moments over the model distribution $p_{\text{ME}}$ we make use of the above factorization:

$$\left\langle\mathbf{x}\mathbf{x}^\top\right\rangle = \left\langle\left\langle\mathbf{x}\mathbf{x}^\top|\mathbf{b}\right\rangle\right\rangle_b = \left(-\mathbf{\Lambda}_{xx}\right)^{-1} + \left\langle\boldsymbol{\mu}_{x|b}\boldsymbol{\mu}_{x|b}^\top\right\rangle_b$$

$$\left\langle\mathbf{x}\mathbf{b}^\top\right\rangle = \left\langle\boldsymbol{\mu}_{x|b}\mathbf{b}^\top\right\rangle_b = \left\langle\mathbf{b}\mathbf{x}^\top\right\rangle^\top, \qquad \left\langle\mathbf{x}\right\rangle = \left\langle\boldsymbol{\mu}_{x|b}\right\rangle_b \tag{9}$$

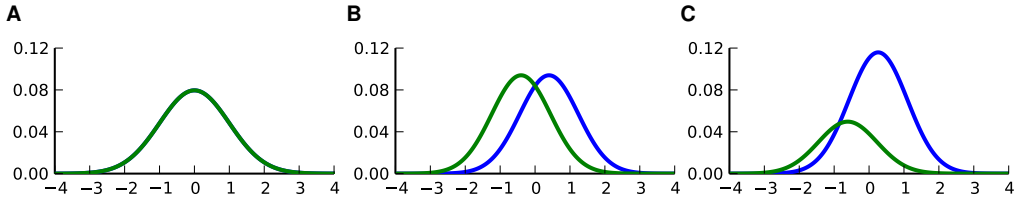

**FIGURE 1:** Illustration of different parameter settings. A:independent binary and continuous variables, B: correlations (0.4) between variables, C: changing mean of the binary variables (here: 0.7) corresponds to changing weightings of the Gaussians, correlations are 0.4. Blue lines indicate $p(x|b=1)$ and green ones $p(x|b=0)$.

Hence, the only average we actually need to evaluate numerically is the one over the binary variables. Unfortunately, we cannot directly set the parameters for the continuous part, as they depend on the ones for the binary part. However, since the above equations can be evaluated analytically, the difficult part is finding the parameters for the binary variables. In particular, if the number of binary variables is large, calculating the partition function can become infeasible. To some extent, this can be remedied by the use of specialized Monte-Carlo algorithms [4].

## 2.1 Example

In order to gain intuition into the properties of the model, we illustrate it in a simple one-dimensional case. From equation (4) for the conditional mean of the continuous variables, we expect the distance between the conditional means $\boldsymbol{\mu}_{x|b}$ to increase with increasing correlation between continuous and binary variables increases. We see that this is indeed the case: While the conditional Gaussians $p(\mathbf{x}|\mathbf{b}=1)$ and $p(\mathbf{x}|\mathbf{b}=0)$ are identical if $x$ and $b$ are uncorrelated (figure 1A), a correlation between $\mathbf{x}$ and $\mathbf{b}$ shifts them away from the unconditional mean (figure 1B). Also, the weight assigned to each of the two Gaussians can be changed. While in figures 1A and 1B $\mathbf{b}$ has a symmetric mean of 0.5, a non-symmetric mean leads to an asymmetry in the weighting of each Gaussian illustrated in figure 1C.

## 2.2 Comparison with other models for the joint modeling of binary and continous data

There are two models in the literature which model the joint distribution of continuous and binary variables, which we will list in the following and compare them to the model derived in this paper.

**Location model**  The location model (LM) [14, 10, 9] also uses the same factorization as above $p(\mathbf{x}, \mathbf{b}) = p(\mathbf{x}|\mathbf{b})p(\mathbf{b})$. However, the distribution for the binary variables $p(\mathbf{b})$ is not of Boltzmann type but a general multinomial distribution and therefore has more degrees of freedom. The conditional distribution $p(\mathbf{x}|\mathbf{b})$ is assumed to be Gaussian with moments $(\boldsymbol{\mu_b}, \boldsymbol{\Sigma_b})$, which can both depend on the conditional state $\mathbf{b}$. Thus to fit the LM usually requires much more data to estimate the moments for every possible binary state. The location model can also be seen as a maximum entropy model in the sense, that it is the distribution with maximal entropy under all distribution with the conditional moments. As fitting this model in its general form is prone to overfitting, various ad hoc constraints have been proposed; see [9] for details.

**Partially dichotomized Gaussian model**  Another simple possibility to obtain a joint distribution of continuous and binary variables is to take multivariate (latent) Gaussian distribution for all variables and then dichotomize those components which should represent the binary variables. Thus, a binary variable $\mathbf{b}_i$ is set to 1 if the underlying Gaussian variables is greater than 0 and it is set to 0 if the Gaussian variable is smaller than 0. This model is known as the partially dichotomized Gaussian (PDG) [6]. Importantly the marginal distribution over the continuous variables is always Gaussian and not a mixture as in our model. The reason for this is that all marginals of a Gaussian distribution are again Gaussian.

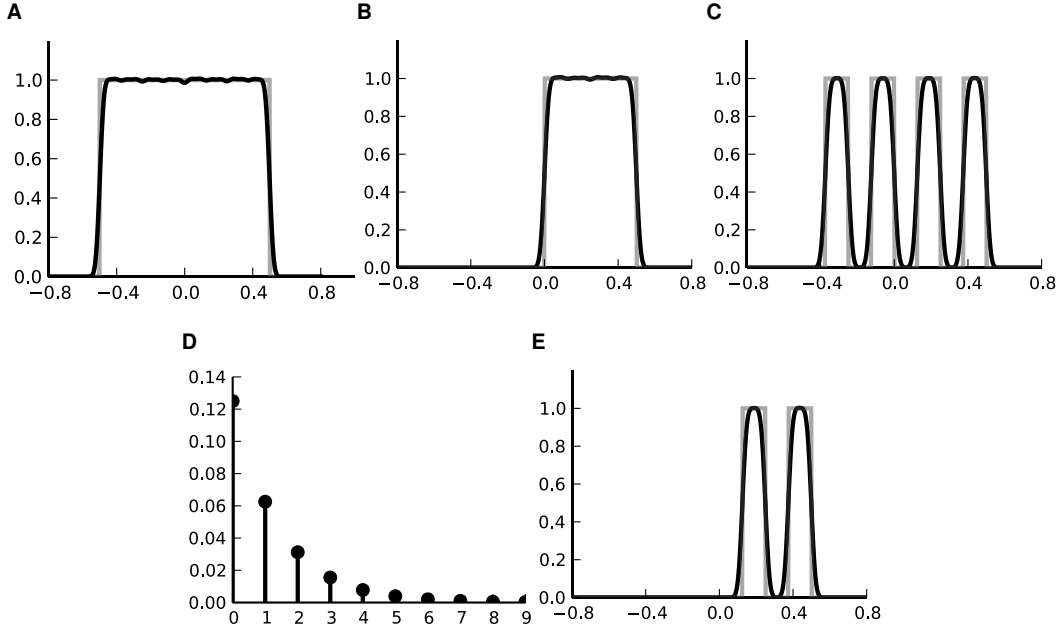

**FIGURE 2:** Illustration of the binary encoding with box-type tuning curves. **A**: shows the marginal distribution over stimuli. The true underlying stimulus distribution is a uniform distribution over the interval ( 0 5 0 5) and is plotted in shaded gray. The mixture of Gaussian approximation of the MaxEnt model is plotted in black. Each neuron has a tuning-curve, consisting of a superposition of box-functions. **B** shows the tuning-curve of the first neuron. This is equivalent to the conditional distribution, when conditioning on the first bit, which indicates if the stimulus is in the right part of the interval. The tuning-curve is a superposition of 5 box-functions. The true tuning curve is plotted in shaded gray whereas the MaxEnt approximation is plotted in black. **C** shows the tuning curve of neuron with index 2. **D**: Covariance between continuous and binary variables as a function of the index of the binary variables. This is the same as the STA for each neuron (see also equation (10)). **E** shows the conditional distribution, when conditioning on both variables (0,2) to be one. This corresponds to the product of the tuning-curves.

## 3 Applications

### 3.1 Spike triggering and feature extraction

Spike triggering is a common technique in order to find features which a single neuron is sensitive to. The presented model can be seen as an extension in the following sense. Suppose that we have observed samples $(\mathbf{x}^n \ \mathbf{b}^n)$ from a population responding to a stimulus. The spike triggered average (STA) for a neuron $i$ is then defined as

$$\text{STA}_i = \frac{\sum_n \mathbf{x}^n \mathbf{b}_i^n}{\sum_n \mathbf{b}_i^n} = \mathbb{E}[\mathbf{x}\mathbf{b}_i]r_i \tag{10}$$

where $r_i = \frac{\sum_n \mathbf{b}_i^n}{N} = p(\mathbf{b}_i = 1)$ is the firing rate of the $i$-th neuron or fraction of ones within the sample. Note, that the moment $\mathbb{E}[\mathbf{x}\mathbf{b}_i]$ is one of the constraints we require for the maximum entropy model and therefore the STA is included in the model.

In addition, the model has also similarities to spike-triggered covariance (STC) [20, 16]. STC denotes the distribution or, more precisely, the covariance of the stimuli that evoked a spiking response. Usually, this covariance is then compared to the total covariance over the entire stimulus distribution. In the joint maximum-entropy model, we have access to a similar distribution, namely the conditional distribution $p(\mathbf{x} \ \mathbf{b}_i = 1)$, which is a compact description of the spike-triggered distribution. Note that $p(\mathbf{x} \ \mathbf{b}_i = 1)$ can be highly non-Gaussian as all neurons $j = i$ are marginalized out – this is why the current model is an extension to spike triggering. Additionally, we can also trigger or con-

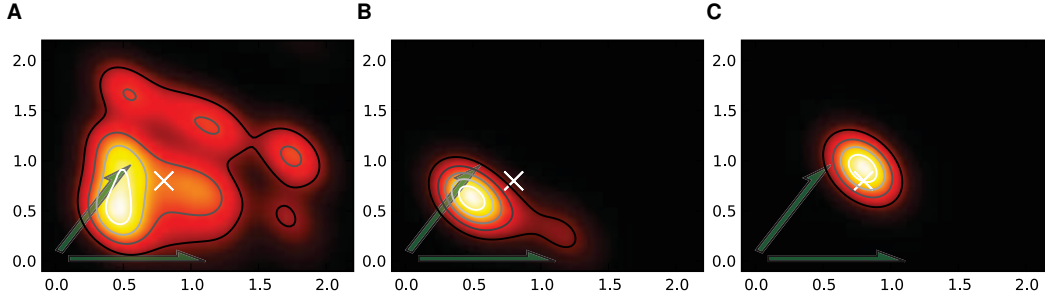

**A**   **B**   **C**

**FIGURE 3:** Illustration of a spike-by-spike decoding scheme. The MaxEnt model was fit to data from two deterministic integrate-and-fire models. The MaxEnt model can then be used for decoding spikes generated by the two independent deterministic models. The two green arrows correspond the weights of a two-pixel receptive field for each of the two neurons. The 2 dimensional stimulus was drawn from two independent Gamma distributions. The resulting spike-trains were discretized in 5 time-bins, each 200 ms long. A spike-train to a particular stimulus ($\mathbf{x}$ cross) is decoded. In **A**) the marginal distribution of the continuous variables is shown. In **B**) the posterior, when conditioning on the first temporal half of the response to that stimulus is shown. Finally in **C**) the conditional distribution, when conditioning on the full observed binary pattern is plotted.

dition not on a single neuron but on any response pattern $\mathbf{B}_{\mathcal{S}}$ of a sub-population $\mathcal{S}$. The resulting $p(\mathbf{x}\,\mathbf{B}_{\mathcal{S}})$ with $\mathbf{B}_{\mathcal{S}} = \mathbf{b}:\mathbf{b}_i = \mathbf{B}_i\ i\ \mathcal{S}$ is then also a mixture of Gaussians with $2^n$ components, where $n$ is the number of unspecified neurons $j\ \mathcal{S}$. As illustrated above (see figure 1B), correlations between neurons and stimuli lead to a separation of the individual Gaussians. Hence, stimulus correlations of other neurons $j = i$ in the distribution $p(\mathbf{x}\,\mathbf{b}_{j=i}\,\mathbf{b}_i = 1)$ would have the same effect on the spike-triggered distribution of neuron $i$. Correlations within this distribution also imply, that there are correlations between neuron $j$ and neuron $i$. Thus, stimulus as well as noise correlations cause deviations of the conditional $p(\mathbf{x}\,\mathbf{B}_{\mathcal{S}})$ from a single Gaussian. Therefore, the full conditional distribution $p(\mathbf{x}\,\mathbf{B}_{\mathcal{S}})$ in general contain more information about the features which trigger this sub-population to evoke the specified response pattern, than the conditional mean, i.e. the STA.

We demonstrate the capabilities of this approach by considering the following encoding. As stimulus, we consider one continuous real valued variable that is drawn uniformly from the interval [ 0 5 0 5]. It is mapped to a binary population response in the following way. Each neuron $i$ has a square-wave tuning function:

$$\mathbf{b}_i(\mathbf{x}) = (\sin{(2\ (i+1)\mathbf{x})})$$

where is the Heaviside function. In this way, the response of a neuron is set to 1 if its tuning-function is positive and 0 otherwise. The first (index 0) neuron distinguishes the left and the right part of the entire interval. The $(i+1)$st neuron distinguishes subsequently left from right in the sub-intervals of the $i$th neuron. That is, the response of the second neuron is always 1, if the stimulus is in the right part of the intervals [ 0 5 0] and [0 0 5]. These tuning curves can also be thought of as a mapping into a non-linear feature space in which the neuron acts linear again. Although the data-generation process is not contained in our model class we were able to extract the tuning curves as shown in figure 2. Note, that for this example neither the STA nor STC analysis alone would provide any insight into the feature selectivity of the neurons, in particular for the neurons which have multi-modal tuning curves (the ones with higher indexes in the above example). However, the tuning curves could be reconstructed with any kind of density estimation, given the STA.

### 3.2 Spike-by-Spike decoding

Since we have a simple expression for the conditional distribution $p(\mathbf{x}\,\mathbf{b}\quad)$ (see equation (4)), we can use the model to analyze the decoding performance of a neural population. To illustrate this, we sampled spike trains from two leaky integrate-and-fire neurons for 1 second and discretized the resulting spike trains into 5 bins of 200 ms length each. Each trial, we used a constant two dimensional stimulus, which was drawn from two independent Gamma distributions with shape

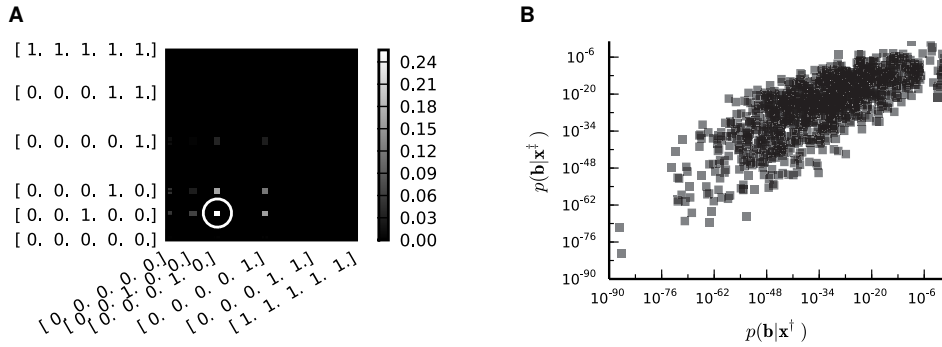

**FIGURE 4:** Illustration of the conditional probability $p(\mathbf{b}|\mathbf{x})$ for the example in figure 3. In 4A, for every binary pattern the corresponding probability is plotted for the given stimulus from figure 3, where the brightness of each square indicates its probability. For the given stimulus the actual response pattern used for figure 3 is marked with a circle. Each pattern $\mathbf{b}$ is split into two halves by the contributions of the two neurons (32 possible patterns for each neuron) and response patterns of the first neuron are shown on the x-axis, while response patterns of the second neuron on the y-axis. In 4B we plotted for each pattern $\mathbf{b}$ its probability under the two conditional distributions $p(\mathbf{b}|\mathbf{x}^\ast)$ and $p(\mathbf{b}|\mathbf{x}^\dagger)$ against each other with $\mathbf{x}^\ast = (0.85\ 0.72)$ and $\mathbf{x}^\dagger = (1.5\ 1.5)$.

parameter $\alpha = 3$ and scale parameter $\beta = 0.3$. For each LIF neuron, this two dimensional stimulus was then projected onto the one-dimensional subspace spanned by its receptive field and used as input current. Hence, there are 10 binary variables, 5 for each spike-train of the neurons and 2 continuous variables for the stimulus to be modeled. We draw $5 \cdot 10^6$ samples, calculated the second order moments of the joint stimulus and response vectors and fitted our maximum entropy model to these moments. The obtained distribution is shown in figure 3. In 3A, we show the marginal distribution of the stimuli, which is a mixture of $2^{10}$ Gaussians. The receptive fields of the two neurons are indicated by green arrows. To illustrate the decoding process, we sampled a stimulus and corresponding response $r$, from which we try to reconstruct the stimulus. In 3B, we show the conditional distribution when conditioning on the first half of the response. Finally in 3C, the complete posterior is shown when conditioned on the full response. From a-c, the posterior is more and more concentrated around the true stimulus. Although there is no neural noise in the encoding model, the reconstruction is not perfect. This is due to the regularization properties of the maximum entropy approach.

### 3.3 Stimulus dependence of firing patterns

While previous studies on the structure of neuronal firing patterns in the retina have compared how well second-order maximum entropy models fit the empirically observed distributions under different stimulation conditions [19, 22], the stimulus has never been explicitly taken into account into the model. In the proposed framework, we have access to $p(\mathbf{b}|\mathbf{x})$, so we can explicitly study how the pattern distribution of a neural population depends on the stimulus. We illustrate this by continuing the example of figure 3. First, we show how the individual firing probabilities depend on $x$ (figure 4A). Note, that although the encoding process for the previous example was noiseless, that is, for every given stimulus there is only one response pattern, the conditional distribution $p(\mathbf{b}|\mathbf{x})$ is not a delta-function, but dispersed around the expected response. This is due to the second order approximation to the encoding model. Further, as it turns out, that a spike in the next bin after a spike is very unlikely under the model, which captures the property of the leaky integrator. Also, we compare how $p(\mathbf{b}|\mathbf{x})$ changes for different values of $\mathbf{x}$. This is illustrated in figure 4B.

### 3.4 Spike train metric

Oftentimes, it is desirable to measure distances between spike trains [24]. One problem, however, is that not every spike might be of equal importance. That is, if a spike train differs only in one spike, it might nevertheless represent a completely different stimulus. Therefore, Ahmadian [2] suggested to measure the distance between spike trains as the difference of stimuli when reconstructed based on

the one or the other spike train seems. If the population is noisy, we want to measure the difference of reconstructed stimuli on average. To this end, we need access to the posterior distribution, when conditioning on a particular spike train or binary pattern. Using the maximum entropy model, we can define the following spike-metric:

$$d(\mathbf{b}^1, \mathbf{b}^2) = D_{\mathrm{KL}}\left[p_{\mathrm{ME}}(\mathbf{x}|\mathbf{b}^1) || p_{\mathrm{ME}}(\mathbf{x}|\mathbf{b}^2)\right] = \frac{1}{2}\left(\left(\boldsymbol{\mu}_{x|b^1} - \boldsymbol{\mu}_{x|b^2}\right)^\top \boldsymbol{\Lambda}_{xx}\left(\boldsymbol{\mu}_{x|b^1} - \boldsymbol{\mu}_{x|b^2}\right)\right) \quad (11)$$

Here, $D_{\mathrm{KL}}$ denotes the Kullback-Leibler divergence between the posterior densities. Equation 11 is symmetric in $\mathbf{b}$, however, in order to get a symmetric expression for other types of posterior distributions, the Jensen-Shannon divergence might be used instead. As an example we consider the induced metrics for the encoding model of figure 2. The metric induced by the square-wave tuning functions of section 3.1 is relatively simple. When conditioning on a particular population response, the conditional distribution $p(\mathbf{x}|\mathbf{b})$ is always a Gaussian with approximately the width of the smallest wavelength. Flipping a neuron's response within this pattern corresponds to shifting the conditional distribution. Suppose we have observed a population response consisting of only ones. This results in a Gaussian posterior distribution with mean in the middle of the rightmost interval $(0.5 - \frac{1}{1024}, 0.5)$. Now flipping the response of the "low-frequency" neuron, that is the one shown in figure 2B, shifts the mean of the posterior to the middle of the sub-interval $(-\frac{1}{1024}, 0)$. Whereas flipping the "high-frequency" neuron, the one which indicates left or right within the smallest possible sub-interval, corresponds to shifting the mean just by the amount of this smallest interval to the left. Flipping the response of single neurons within this population can result in posterior distribution which look quite different in terms of the Kullback-Leibler divergence. In particular, there is an ordering in terms of the frequency of the neurons with respect to the proposed metric.

## 4    Conclusion

We have presented a maximum-entropy model based on the joint second order statistics of continuous valued variables and binary neural responses. This allows us to extend the maximum-entropy approach [19] for analyzing neural data to incorporate other variables of interest such as continuous valued stimuli. Alternatively, additional neurophysiological signals such as local field potentials [13] can be taken into account to study their relation with the joint firing patterns of local neural ensembles. We have demonstrated four applications of this approach: (1) It allows us to extract the features a (sub-)population of neurons is sensitive to, (2) we can use it for spike-by-spike decoding, (3) we can assess the impact of stimuli on the distribution of population patterns and (4) it yields a natural spike-train metric.

We have shown that the joint maximum-entropy model can be learned in a convex fashion, although high-dimensional binary patterns might require the use of efficient sampling techniques. Because of the maximum-entropy approach the resulting distribution is well regularized and does not require any ad-hoc restrictions or regularity assumptions as have been proposed for related models [9]. Analogous to a Boltzmann machine with hidden variables, it is possible to further add hidden binary nodes to the model. This allows us to take higher-order correlations into account as well, although we stay essentially in the second-order framework. Fortunately, the learning scheme for fitting the modified model to observed data remains almost unchanged: The only difference is that the moments have to be averaged over the non-observed binary variables as well. In this way, the model can also be used as a clustering algorithm if we marginalize over all binary variables. The resulting mixture of Gaussian model will consist of $2^N$ components, where $N$ is the number of hidden binary variables. Unfortunately, convexity cannot be guaranteed if the model contains hidden nodes. In a similar fashion, we could also add hidden continuous variables, for example to model unobserved common inputs. In contrast to hidden binary nodes, this does not lead to an increased model complexity: averaging over hidden continuous variables corresponds to integrating out each Gaussian within the mixture, which results in another Gaussian. Also the restriction that all covariance matrices in the mixture need to be the same still holds, because each Gaussian is integrated in the same way.

**Acknowledgments**    We would like to thank J. Macke and J. Cotton for discussions and feedback on the manuscript. This work is supported by the German Ministry of Education, Science, Research and Technology through the Bernstein award to MB (BMBF; FKZ: 01GQ0601), the Werner-Reichardt Centre for Integrative Neuroscience Tübingen, and the Max Planck Society.

# References

[1] D.H. Ackley, G.E. Hinton, and T.J. Sejnowski. A learning algorithm for boltzmann machines. *Cognitive Science*, 9:147–169, 1985.

[2] Y. Ahmadian, J. Pillow, J. Shlens, E. Simoncelli, E.J. Chichilinsky, and L. Paninski. A decoder-based spike train metric for analyzing the neural code in the retina. In *Frontiers in Systems Neuroscience. Conference Abstract: Computational and systems neuroscience*, 2009.

[3] M. Bethge and P. Berens. Near-Maximum entropy models for binary neural representations of natural images. In *Proceedings of the Twenty-First Annual Conference on Neural Information Processing Systems*, volume 20, pages 97–104, Cambridge, MA, 2008. MIT Press.

[4] Tamara Broderick, Miroslav Dudik, Gasper Tkacik, Robert E Schapire, and William Bialek. Faster solutions of the inverse pairwise ising problem. *arXiv*, q-bio.QM:0712.2437, Dec 2007.

[5] G. Buzsaki. Large-scale recording of neuronal ensembles. *Nature Neuroscience*, 7(5):446–451, 2004.

[6] D. R. Cox and Nanny Wermuth. Likelihood factorizations for mixed discrete and continuous variables. *Scandinavian Journal of Statistics*, 26(2):209–220, June 1999.

[7] A. Tang et al. A maximum entropy model applied to spatial and temporal correlations from cortical networks in vitro. *J. Neurosci.*, 28(2):505–518, 2008.

[8] Q.J.M. Huys, M.B. Ahrens, and L. Paninski. Efficient estimation of detailed single-neuron models. *Journal of neurophysiology*, 96(2):872, 2006.

[9] W. Krzanowski. The location model for mixtures of categorical and continuous variables. *Journal of Classification*, 10(1):25–49, 1993.

[10] S. L. Lauritzen and N. Wermuth. Graphical models for associations between variables, some of which are qualitative and some quantitative. *The Annals of Statistics*, 17(1):31–57, March 1989.

[11] D.J.C. MacKay. *Information theory, inference and learning algorithms*. Cambridge U. Press, 2003.

[12] J.H. Macke, P. Berens, A.S. Ecker, A.S. Tolias, and M. Bethge. Generating spike trains with specified correlation coefficients. *Neural Computation*, 21(2):1–27, 2009.

[13] Marcelo A. Montemurro, Malte J. Rasch, Yusuke Murayama, Nikos K. Logothetis, and Stefano Panzeri. Phase-of-Firing coding of natural visual stimuli in primary visual cortex. *Current Biology*, Vol 18:375–380, March 2008.

[14] I. Olkin and R. F. Tate. Multivariate correlation models with mixed discrete and continuous variables. *The Annals of Mathematical Statistics*, 32(2):448–465, June 1961.

[15] J.W. Pillow, J. Shlens, L. Paninski, A. Sher, A.M. Litke, EJ Chichilnisky, and E.P. Simoncelli. Spatio-temporal correlations and visual signalling in a complete neuronal population. *Nature*, 454(7207):995–999, 2008.

[16] J.W. Pillow and E.P. Simoncelli. Dimensionality reduction in neural models: an information-theoretic generalization of spike-triggered average and covariance analysis. *Journal of Vision*, 6(4):414–428, 2006.

[17] Y. Roudi, E. Aurell, and J.A. Hertz. Statistical physics of pairwise probability models. *Frontiers in Computational Neuroscience*, 2009.

[18] Yasser Roudi, Sheila Nirenberg, and Peter E. Latham. Pairwise maximum entropy models for studying large biological systems: When they can work and when they can't. *PLoS Comput Biol*, 5(5), 2009.

[19] Elad Schneidman, Michael J. Berry, Ronen Segev, and William Bialek. Weak pairwise correlations imply strongly correlated network states in a neural population. *Nature*, 440(7087):1007–1012, April 2006.

[20] O. Schwartz, EJ Chichilnisky, and E.P. Simoncelli. Characterizing neural gain control using spike-triggered covariance. In *Advances in Neural Information Processing Systems 14: Proceedings of the 2002 [sic] Conference*, page 269. MIT Press, 2002.

[21] J. Shlens, G. D. Field, J. L. Gauthier, M. Greschner, A. Sher, A. M. Litke, and E. J. Chichilnisky. The structure of Large-Scale synchronized firing in primate retina. *Journal of Neuroscience*, 29(15):5022, 2009.

[22] Jonathon Shlens, Greg D. Field, Jeffrey L. Gauthier, Matthew I. Grivich, Dumitru Petrusca, Alexander Sher, Alan M. Litke, and E. J. Chichilnisky. The structure of Multi-Neuron firing patterns in primate retina. *J. Neurosci.*, 26(32):8254–8266, August 2006.

[23] Jonathon Shlens, Fred Rieke, and E. J. Chichilnisky. Synchronized firing in the retina. *Current Opinion in Neurobiology*, 18(4):396–402, August 2008.

[24] J.D. Victor and K.P. Purpura. Metric-space analysis of spike trains: theory, algorithms and application. *Network: computation in neural systems*, 8(2):127–164, 1997.

[25] Shan Yu, Debin Huang, Wolf Singer, and Danko Nikolic. A small world of neuronal synchrony. *Cereb. Cortex*, 18(12):2891–2901, April 2008.

